# Dirty Statistical Models

**Eunho Yang**
Department of Computer Science
University of Texas at Austin
eunho@cs.utexas.edu

**Pradeep Ravikumar**
Department of Computer Science
University of Texas at Austin
pradeepr@cs.utexas.edu

## Abstract

We provide a unified framework for the high-dimensional analysis of "superposition-structured" or "dirty" statistical models: where the model parameters are a *superposition* of structurally constrained parameters. We allow for any number and types of structures, and any statistical model. We consider the general class of $M$-estimators that minimize the sum of any loss function, and an instance of what we call a "hybrid" regularization, that is the infimal convolution of weighted regularization functions, one for each structural component. We provide corollaries showcasing our unified framework for varied statistical models such as linear regression, multiple regression and principal component analysis, over varied superposition structures.

## 1   Introduction

High-dimensional statistical models have been the subject of considerable focus over the past decade, both theoretically as well as in practice. In these high-dimensional models, the ambient dimension of the problem $p$ may be of the same order as, or even substantially larger than the sample size $n$. It has now become well understood that even in this type of high-dimensional $p \gg n$ scaling, it is possible to obtain statistically consistent estimators provided one imposes structural constraints on the statistical models. Examples of such structural constraints include sparsity constraints (e.g. compressed sensing), graph-structure (for graphical model estimation), low-rank structure (for matrix-structured problems), and sparse additive structure (for non-parametric models), among others. For each of these structural constraints, a large body of work have proposed and analyzed statistically consistent estimators. For instance, a key subclass leverage such structural constraints via specific regularization functions. Examples include $\ell_1$-regularization for sparse models, nuclear norm regularization for low-rank matrix-structured models, and so on.

A caveat to this strong line of work is that imposing such "clean" structural constraints such as sparsity or low-rank structure, is typically too stringent for real-world messy data. What if the parameters are not exactly sparse, or not exactly low rank? Indeed, over the last couple of years, there has been an emerging line of work that address this caveat by "mixing and matching" different structures. Chandrasekaran et al. [5] consider the problem of recovering an unknown low-rank and an unknown sparse matrix, given the sum of the two matrices; for which they point to applications in system identification in linear time-invariant systems, and optical imaging systems among others. Chandrasekaran et al. [6] also apply this matrix decomposition estimation to the learning of latent-variable Gaussian graphical models, where they estimate an inverse covariance matrix that is the sum of sparse and low-rank matrices. A number of papers have applied such decomposition estimation to robust principal component analysis: Candès et al. [3] learn a covariance matrix that is the sum of a low-rank factored matrix and a sparse "error/outlier" matrix, while [9, 15] learn a covariance matrix that is the sum of a low-rank matrix and a column-sparse error matrix. Hsu et al. [7] analyze this estimation of a sum of a low-rank and elementwise sparse matrix in the noisy setting; while Agarwal et al. [1] extend this to the sum of a low-rank matrix and a matrix with general structure. Another application is multi-task learning, where [8] learn a multiple-linear-regression coefficient

matrix that is the sum of a sparse and a block-sparse matrix. This strong line of work can be seen to follow the resume of estimating a superposition of two structures; and indeed their results show this simple extension provides a vast increase in the practical applicability of structurally constrained models. The statistical guarantees in these papers for the corresponding $M$-estimators typically require fairly extensive technical arguments that extend the analyses of specific *single-structured* regularized estimators in highly non-trivial ways.

This long-line of work above on $M$-estimators and analyses for specific pairs of super-position structures for specific statistical models, lead to the question: is there a unified framework for studying *any general* tuple (i.e. not just a pair) of structures, for any *general statistical model*? This is precisely the focus of this paper: we provide a unified framework of "superposition-structured" or "dirty" statistical models, with any number and any types of structures, for any statistical model. By such "superposition-structure," we mean the constraint that the parameter be a *superposition* of "clean" structurally constrained parameters. In addition to the motivation above, of unifying the burgeoning list of works above, as well as to provide guarantees for many novel superpositions (of for instance more than two structures) not yet considered in the literature; another key motivation is to provide insights on the key ingredients characterizing the statistical guarantees for such dirty statistical models. Our unified analysis allows the following very general class of $M$-estimators, which are the sum of *any loss function*, and an instance of what we call a "hybrid" regularization function, that is the infimal convolution of any weighted regularization functions, one for each structural component. As we show, this is equivalent to an $M$-estimator that is the sum of (a) a loss function applied to the sum of the multiple parameter vectors, one corresponding to each structural component; and (b) a weighted sum of regularization functions, one for each of the parameter vectors. We stress that our analysis allows for *general loss functions*, and *general component regularization functions*. We provide corollaries showcasing our unified framework for varied statistical models such as linear regression, multiple regression and principal component analysis, over varied superposition structures.

## 2  Problem Setup

We consider the following general statistical modeling setting. Consider a random variable $Z$ with distribution $\mathbb{P}$, and suppose we are given $n$ observations $Z_1^n := \{Z_1, \ldots, Z_n\}$ drawn i.i.d. from $\mathbb{P}$. We are interested in estimating some parameter $\theta^* \in \mathbb{R}^p$ of the distribution $\mathbb{P}$. We assume that the statistical model parameter $\theta^*$ is "superposition-structured," so that it is the sum of parameter components, each of which is constrained by a specific structure. For a formalization of the notion of structure, we first review some terminology from [11]. There, they use subspace pairs $(\mathcal{M}, \overline{\mathcal{M}}^\perp)$, where $\mathcal{M} \subseteq \overline{\mathcal{M}}$, to capture any structured parameter. $\mathcal{M}$ is the *model subspace* that captures the constraints imposed on the model parameter, and is typically *low-dimensional*. $\overline{\mathcal{M}}^\perp$ is the *perturbation subspace* of parameters that represents perturbations away from the model subspace. They also define the property of *decomposability* of a regularization function, which captures the suitablity of a regularization function $\mathcal{R}$ to particular structure. Specifically, a regularization function $\mathcal{R}$ is said to be *decomposable* with respect to a subspace pair $(\mathcal{M}, \overline{\mathcal{M}}^\perp)$, if

$$\mathcal{R}(u + v) = \mathcal{R}(u) + \mathcal{R}(v), \quad \text{for all } u \in \mathcal{M}, \ v \in \overline{\mathcal{M}}^\perp.$$

For any structure such as sparsity, low-rank, etc., we can define the corresponding low-dimensional model subspaces, as well as regularization functions that are decomposable with respect to the corresponding subspace pairs.

**I.** *Sparse vectors.* Given any subset $S \subseteq \{1, \ldots, p\}$ of the coordinates, let $\mathcal{M}(S)$ be the *subspace* of vectors in $\mathbb{R}^p$ that have support contained in $S$. It can be seen that any parameter $\theta \in \mathcal{M}(S)$ would be atmost $|S|$-sparse. For this case, we use $\overline{\mathcal{M}}(S) = \mathcal{M}(S)$, so that $\overline{\mathcal{M}}^\perp(S) = \mathcal{M}^\perp(S)$. As shown in [11], the $\ell_1$ norm $\mathcal{R}(\theta) = \|\theta\|_1$, commonly used as a sparsity-encouraging regularization function, is decomposable with respect to subspace pairs $(\mathcal{M}(S), \overline{\mathcal{M}}^\perp(S))$.

**II.** *Low-rank matrices.* Consider the class of matrices $\Theta \in \mathbb{R}^{k \times m}$ that have rank $r \leq \min\{k, m\}$. For any given matrix $\Theta$, we let $\text{row}(\Theta) \subseteq \mathbb{R}^m$ and $\text{col}(\Theta) \subseteq \mathbb{R}^k$ denote its row space and column space respectively. For a given pair of $r$-dimensional subspaces $U \subseteq \mathbb{R}^k$ and $V \subseteq \mathbb{R}^m$, we define the subspace pairs as follows: $\mathcal{M}(U, V) := \left\{ \Theta \in \mathbb{R}^{k \times m} \mid \text{row}(\Theta) \subseteq V, \ \text{col}(\Theta) \subseteq U \right\}$ and

$\overline{\mathcal{M}}^\perp(U, V) := \{\Theta \in \mathbb{R}^{k \times m} \mid \text{row}(\Theta) \subseteq V^\perp, \ \text{col}(\Theta) \subseteq U^\perp\}$. As [11] show, the nuclear norm $\mathcal{R}(\theta) = \|\theta\|_1$ is decomposable with respect to the subspace pairs $(\mathcal{M}(U, V), \overline{\mathcal{M}}^\perp(U, V))$.

In our dirty statistical model setting, we do not just have one, but a set of structures; suppose we index them by the set $I$. Our key structural constraint can then be stated as: $\theta^* = \sum_{\alpha \in I} \theta_\alpha^*$, where $\theta_\alpha^*$ is a "clean" structured parameter with respect to a subspace pair $(\mathcal{M}_\alpha, \overline{\mathcal{M}}_\alpha^\perp)$, for $\mathcal{M}_\alpha \subseteq \overline{\mathcal{M}}_\alpha$. We also assume we are given a set of regularization functions $\mathcal{R}_\alpha(\cdot)$, for $\alpha \in I$ that are suited to the respective structures, in the sense that they are decomposable with respect to the subspace pairs $(\mathcal{M}_\alpha, \overline{\mathcal{M}}_\alpha^\perp)$.

Let $\mathcal{L} : \Omega \times \mathcal{Z}^n \mapsto \mathbb{R}$ be some loss function that assigns a cost to any parameter $\theta \in \Omega \subseteq \mathbb{R}^p$, for a given set of observations $Z_1^n$. For ease of notation, in the sequel, we adopt the shorthand $\mathcal{L}(\theta)$ for $\mathcal{L}(\theta; Z_1^n)$. We are interested in the following "super-position" estimator:

$$\min_{(\theta_\alpha)_{\alpha \in I}} \mathcal{L}\left(\sum_{\alpha \in I} \theta_\alpha\right) + \sum_{\alpha \in I} \lambda_\alpha \, \mathcal{R}_\alpha(\theta_\alpha), \tag{1}$$

where $(\lambda_\alpha)_{\alpha \in I}$ are the regularization penalties. This optimization problem involves not just one parameter vector, but multiple parameter vectors, one for each structural component: while the loss function applies only to the sum of these, separate regularization functions are applied to the corresponding parameter vectors. We will now see that this can be re-written to a standard $M$-estimation problem which minimizes, over a *single parameter vector*, the sum of a loss function and a special "dirty" regularization function.

Given a vector $\mathbf{c} := (c_\alpha)_{\alpha \in I}$ of convex-combination weights, suppose we define the following "dirty" regularization function, that is the infimal convolution of a set of regularization functions:

$$\mathcal{R}(\theta; \mathbf{c}) = \inf\left\{\sum_{\alpha \in I} c_\alpha \mathcal{R}_\alpha(\theta_\alpha) : \sum_{\alpha \in I} \theta_\alpha = \theta\right\}. \tag{2}$$

It can be shown that provided the individual regularization functions $\mathcal{R}_\alpha(\cdot)$, for $\alpha \in I$, are *norms*, $\mathcal{R}(\cdot; \mathbf{c})$ is a norm as well. We discuss this and other properties of this hybrid regularization function $\mathcal{R}(\cdot; \mathbf{c})$ in Appendix A.

**Proposition 1.** *Suppose $(\widehat{\theta}_\alpha)_{\alpha \in I}$ is the solution to the $M$-estimation problem in* (1)*. Then $\widehat{\theta} := \sum_{\alpha \in I} \widehat{\theta}_\alpha$ is the solution to the following problem:*

$$\min_{\theta \in \Omega} \mathcal{L}(\theta) + \lambda \mathcal{R}(\theta; \mathbf{c}), \tag{3}$$

*where $c_\alpha = \lambda_\alpha / \lambda$. Similarly, if $\widehat{\theta}$ is the solution to* (3)*, then there is a solution $(\widehat{\theta}_\alpha)_{\alpha \in I}$ to the $M$-estimation problem* (1)*, such that $\widehat{\theta} := \sum_{\alpha \in I} \widehat{\theta}_\alpha$.*

Proposition 1 shows that the optimization problems (1) and (3) are equivalent. While the tuning parameters in (1) correspond to the regularization penalties $(\lambda_\alpha)_{\alpha \in I}$, the tuning parameters in (3) correspond to the weights $(c_\alpha)_{\alpha \in I}$ specifying the "dirty" regularization function. In our unified analysis theorem, we will provide guidance on setting these tuning parameters as a function of various model-parameters.

## 3 Error Bounds for Convex $M$-estimators

Our goal is to provide error bounds $\|\widehat{\theta} - \theta^*\|$, between the target parameter $\theta^*$, the minimizer of the population risk, and our $M$-estimate $\widehat{\theta}$ from (1), for *any error norm* $\|\cdot\|$. A common example of an error norm for instance is the $\ell_2$ norm $\|\cdot\|_2$. We now turn to the properties of the loss function and regularization function that underlie our analysis. We first restate some natural assumptions on the loss and regularization functions.

**(C1)** The loss function $\mathcal{L}$ is convex and differentiable.

**(C2)** The regularizers $\mathcal{R}_\alpha$ are norms, and are decomposable with respect to the subspace pairs $(\mathcal{M}_\alpha, \overline{\mathcal{M}}_\alpha^\perp)$, where $\mathcal{M}_\alpha \subseteq \overline{\mathcal{M}}_\alpha$.

Our next assumption is a *restricted strong convexity* assumption [11]. Specifically, we will require the loss function $\mathcal{L}$ to satisfy:

**(C3) (Restricted Strong Convexity)** For all $\Delta_\alpha \in \Omega_\alpha$, where $\Omega_\alpha$ is the parameter space for the parameter component $\alpha$,

$$\delta\mathcal{L}(\Delta_\alpha; \theta^*) := \mathcal{L}(\theta^* + \Delta_\alpha) - \mathcal{L}(\theta^*) - \langle\nabla_\theta\mathcal{L}(\theta^*), \Delta_\alpha\rangle \geq \kappa_\mathcal{L}\|\Delta_\alpha\|^2 - g_\alpha\mathcal{R}_\alpha^2(\Delta_\alpha),$$

where $\kappa_\mathcal{L}$ is a "curvature" parameter, and $g_\alpha\mathcal{R}_\alpha^2(\Delta_\alpha)$ is a "tolerance" parameter.

Note that these conditions (C1)-(C3) are imposed even when the model has a single clean structural constraint; see [11]. Note that $g_\alpha$ is usually a function on the problem size decreasing in the sample size; in the standard Lasso with $|I| = 1$ for instance, $g_\alpha = \frac{\log p}{n}$.

Our next assumption is on the interaction between the different structured components.

**(C4) (Structural Incoherence)** For all $\Delta_\alpha \in \Omega_\alpha$,

$$\left|\mathcal{L}\left(\theta^* + \sum_{\alpha \in I}\Delta_\alpha\right) + (|I| - 1)\mathcal{L}(\theta^*) - \sum_{\alpha \in I}\mathcal{L}\left(\theta^* + \Delta_\alpha\right)\right| \leq \frac{\kappa_\mathcal{L}}{2}\sum_{\alpha \in I}\|\Delta_\alpha\|^2 + \sum_{\alpha \in I}h_\alpha\mathcal{R}_\alpha^2(\Delta_\alpha).$$

Note that for a model with a single clean structural constraint, with $|I| = 1$, the condition (C4) is trivially satisfied since the LHS becomes 0. We will see in the sequel that for a large collection of loss functions including all linear loss functions, the condition (C4) simplifies considerably, and moreover holds with high probability, typically with $h_\alpha = 0$. We note that this condition is much weaker than "incoherence" conditions typically imposed when analyzing specific instances of such superposition-structured models (see e.g. references in the introduction), where the assumptions typically include (a) assuming that the structured subspaces $(\mathcal{M}_\alpha)_{\alpha \in I}$ intersect only at $\{0\}$, and (b) that the sizes of these subspaces are extremely small.

Finally, we will use the notion of *subspace compatibility constant* defined in [11], that captures the relationship between the regularization function $\mathcal{R}(\cdot)$ and the error norm $\|\cdot\|$, over vectors in the subspace $\mathcal{M}$: $\Psi(\mathcal{M}, \|\cdot\|) := \sup_{u \in \mathcal{M}\backslash\{0\}}\frac{\mathcal{R}}{\|u\|}$.

**Theorem 1.** *Suppose we solve the $M$-estimation problem in* (3)*, with hybrid regularization* $\mathcal{R}(\cdot; \mathbf{c})$*, where the convex-combination weights* $\mathbf{c}$ *are set as* $c_\alpha = \lambda_\alpha / \sum_{\alpha \in I}\lambda_\alpha$*, with* $\lambda_\alpha \geq 2\mathcal{R}_\alpha^*\left(\nabla_{\theta_\alpha}\mathcal{L}(\theta^*; Z_1^n)\right)$*. Further, suppose conditions (C1) - (C4) are satisfied. Then, the parameter error bounds are given as:*

$$\|\widehat{\theta} - \theta^*\| \leq \left(\frac{3|I|}{2\bar{\kappa}}\right)\max_{\alpha \in I}\lambda_\alpha\Psi_\alpha(\overline{\mathcal{M}}_\alpha) + (|I|\sqrt{\tau_\mathcal{L}}/\sqrt{\bar{\kappa}}),$$

*where*

$$\bar{\kappa} := \frac{\kappa_\mathcal{L}}{2} - 32\bar{g}^2|I|\left(\max_{\alpha \in I}\lambda_\alpha\Psi_\alpha(\overline{\mathcal{M}}_\alpha)\right)^2, \quad \bar{g} := \max_\alpha\frac{1}{\lambda_\alpha}\sqrt{g_\alpha + h_\alpha},$$

$$\tau_\mathcal{L} := \sum_{\alpha \in I}\left[32\bar{g}^2\lambda_\alpha^2\mathcal{R}_\alpha^2\left(\Pi_{\mathcal{M}_\alpha^\perp}(\theta_\alpha^*)\right) + \frac{2\lambda_\alpha}{|I|}\mathcal{R}_\alpha\left(\Pi_{\mathcal{M}_\alpha^\perp}(\theta_\alpha^*)\right)\right].$$

**Remarks: (R1)** It is instructive to compare Theorem 1 to the main Theorem in [11], where they derive parameter error bounds for any $M$-estimator with a decomposable regularizer, for any "clean" structure. Our theorem can be viewed as a generalization: we recover their theorem when we have a single structure with $|I| = 1$. We cannot derive our result in turn from their theorem applied to the $M$-estimator (3) with the hybrid regularization function $\mathcal{R}(\cdot; \mathbf{c})$: the "superposition" structure is not captured by a pair of subspaces, nor is the hybrid regularization function decomposable, as is required by their theorem. Our setting as well as analysis is strictly more general, because of which we needed the additional structural incoherence assumption (C4) (which is trivially satisfied when $|I| = 1$).

**(R2)** Agarwal et al. [1] provide Frobenius norm error bounds for the matrix-decomposition problem of recovering the sum of low-rank and a general structured matrix. In addition to the greater generality of our theorem and framework, Theorem 1 addresses two key drawbacks of their theorem even in their specific setting. First, the proof for their theorem requires the regularization

penalty $\lambda$ for the second structure to be strongly bounded away from zero: their convergence rate does not approach zero even with infinite number of samples $n$. Theorem 1, in contrast, imposes the weaker condition $\lambda_\alpha \geq 2\mathcal{R}_\alpha^*\big(\nabla_{\theta_\alpha}\mathcal{L}(\theta^*; Z_1^n)\big)$, which as we show in the corollaries, allows for the convergence rates to go to zero as a function of the samples. Second, they assumed much stronger conditions for their theorem to hold; in Theorem 1 in contrast, we pose much milder "local" RSC conditions (C3), and a structural incoherence condition (C4).

**(R3)** The statement in the theorem is *deterministic* for fixed choices of $(\lambda_\alpha)$. We also note that the theorem holds for any set of subspace pairs $(\mathcal{M}_\alpha, \overline{\mathcal{M}}_\alpha^\perp)_{\alpha \in I}$ with respect to which the corresponding regularizers are decomposable. As noted earlier, the $\mathcal{M}_\alpha$ should ideally be set to the structured subspace in which the true parameter at least approximately lies, and which we want to be as small as possible (note that the bound includes a term that depends on the size of this subspace via the subspace compatibility constant). In particular, if we assume that the subspaces are chosen so that $\Pi_{\mathcal{M}_\alpha^\perp}(\theta_\alpha^*) = 0$ i.e. $\theta_\alpha^* \in \mathcal{M}_\alpha$, then we obtain the simpler bound in the following corollary.

**Corollary 1.** *Suppose we solve the $M$-estimation problem in* (1)*, with hybrid regularization $\mathcal{R}(\cdot; \mathbf{c})$, where the convex-combination weights $\mathbf{c}$ are set as $c_\alpha = \lambda_\alpha / \sum_{\alpha \in I} \lambda_\alpha$, with $\lambda_\alpha \geq 2\mathcal{R}_\alpha^*\big(\nabla_{\theta_\alpha}\mathcal{L}(\theta^*; Z_1^n)\big)$, and suppose conditions (C1) - (C4) are satisfied. Further, suppose that the subspace-pairs are chosen so that $\theta_\alpha^* \in \mathcal{M}_\alpha$. Then, the parameter error bounds are given as:*

$$\|\widehat{\theta} - \theta^*\| \leq \left(\frac{3|I|}{2\overline{\kappa}}\right)\max_{\alpha \in I}\lambda_\alpha\Psi_\alpha(\overline{\mathcal{M}}_\alpha).$$

It is now instructive to compare the bounds of Theorem 1, and Corollary 1. Theorem 1 has two terms: the first of which is the sole term in the bound in Corollary 1. This first term can be thought of as the "estimation error" component of the error bound, when the parameter has exactly the structure being modeled by the regularizers. The second term can be thought of as the "approximation error" component of the error bound, which is the penalty for the parameter not exactly lying in the structured subspaces modeled by the regularizers. The key term in the "estimation error" component, in Theorem 1, and Corollary 1, is:

$$\Phi = \max_{\alpha \in I}\lambda_\alpha\Psi_\alpha(\overline{\mathcal{M}}_\alpha).$$

Note that each $\lambda_\alpha$ is larger than a particular norm of the sample score function (gradient of the loss at the true parameter): since the expected value of the score function is zero, the magnitude of the sample score function captures the amount of "noise" in the data. This is in turn scaled by $\Psi_\alpha(\overline{\mathcal{M}}_\alpha)$, which captures the size of the structured subspace corresponding to the parameter component $\theta_\alpha^*$. $\Phi$ can thus be thought of as capturing the amount of noise in the data *relative* to the particular structure at hand.

We now provide corollaries showcasing our unified framework for varied statistical models such as linear regression, multiple regression and principal component analysis, over varied superposition structures.

## 4 Convergence Rates for Linear Regression

In this section, we consider the linear regression model:

$$Y = X\theta^* + w, \tag{4}$$

where $Y \in \mathbb{R}^n$ is the observation vector, and $\theta^* \in \mathbb{R}^p$ is the true parameter. $X \in \mathbb{R}^{n \times p}$ is the "observation" matrix; while $w \in \mathbb{R}^n$ is the observation noise. For this class of statistical models, we will consider the instantiation of (1) with the loss function $\mathcal{L}$ consisting of the squared loss:

$$\min_{(\theta_\alpha)_{\alpha \in I}}\left\{\frac{1}{n}\Big\|Y - X\big(\sum_{\alpha \in I}\theta_\alpha\big)\Big\|_2^2 + \sum_{\alpha \in I}\lambda_\alpha\,\mathcal{R}_\alpha(\theta_\alpha)\right\}. \tag{5}$$

For this regularized least squared estimator (5), conditions (C1-C2) in Theorem 1 trivially hold. The *restricted strong convexity* condition (C3) reduces to the following. Noting that $\mathcal{L}(\theta^* + \Delta_\alpha) - \mathcal{L}(\theta^*) - \langle\nabla_\theta\mathcal{L}(\theta^*), \Delta_\alpha\rangle = \frac{1}{n}\|X\Delta_\alpha\|_2^2$, we obtain the following *restricted eigenvalue* condition:

**(D3)** $\frac{1}{n}\|X\Delta_\alpha\|_2^2 \geq \kappa_{\mathcal{L}}\|\Delta_\alpha\|^2 - g_\alpha \mathcal{R}_\alpha^2(\Delta_\alpha)$ for all $\Delta_\alpha \in \Omega_\alpha$.

Finally, our *structural incoherence* condition reduces to the following: Noting that $\big|\mathcal{L}(\theta^* + \sum_{\alpha\in I}\Delta_\alpha) + (|I| - 1)\mathcal{L}(\theta^*) - \sum_{\alpha\in I}\mathcal{L}(\theta^* + \Delta_\alpha)\big| = \frac{2}{n}\big|\sum_{\alpha<\beta}\langle X\Delta_\alpha, X\Delta_\beta\rangle\big|$ in this specific case,

**(D4)** $\frac{2}{n}\big|\sum_{\alpha<\beta}\langle X\Delta_\alpha, X\Delta_\beta\rangle\big| \leq \frac{\kappa_{\mathcal{L}}}{2}\sum_{\alpha\in I}\|\Delta_\alpha\|^2 + \sum_{\alpha\in I} h_\alpha \mathcal{R}_\alpha^2(\Delta_\alpha)$.

### 4.1 Structural Incoherence with Gaussian Design

We now show that the condition (D4) required for Theorem 1, holds with high probability when the observation matrix is drawn from a so-called $\Sigma$-*Gaussian ensemble*: where each row $X_i$ is independently sampled from $N(0, \Sigma)$. Before doing so, we first state some assumption on the population covariance matrix $\Sigma$. Let $\mathcal{P}_M$ denote the matrix corresponding to the projection operator for the subspace $M$. We will then require the following assumption:

**(C-Linear)** Let $\Lambda := \max_{\gamma_1, \gamma_2}\left\{2 + \frac{3\lambda_{\gamma_1}\Psi_{\gamma_1}(\bar{\mathcal{M}}_{\gamma_1})}{\lambda_{\gamma_2}\Psi_{\gamma_2}(\mathcal{M}_{\gamma_2})}\right\}$. For any $\alpha, \beta \in I$,

$$\max\left\{\sigma_{\max}\left(\mathcal{P}_{\bar{\mathcal{M}}_\alpha}\Sigma\mathcal{P}_{\bar{\mathcal{M}}_\beta}\right), \sigma_{\max}\left(\mathcal{P}_{\bar{\mathcal{M}}_\alpha}\Sigma\mathcal{P}_{\bar{\mathcal{M}}_\beta^\perp}\right), \sigma_{\max}\left(\mathcal{P}_{\bar{\mathcal{M}}_\alpha^\perp}\Sigma\mathcal{P}_{\bar{\mathcal{M}}_\beta^\perp}\right)\right\} \leq \frac{\kappa_{\mathcal{L}}}{8\binom{|I|}{2}\Lambda^2|I|}. \quad (6)$$

**Proposition 2.** *Suppose each row $X_i$ of the observation matrix $X$ is independently sampled from $N(0, \Sigma)$, and the condition (C-Linear) (6) holds. Further, suppose that $\Pi_{\mathcal{M}_\alpha^\perp}(\theta_\alpha^*) = 0$, for all $\alpha \in I$. Then, it holds that with probability at least $1 - \frac{4}{\max\{n,p\}}$,*

$$\frac{2}{n}\big|\sum_{\alpha<\beta}\langle X\Delta_\alpha, X\Delta_\beta\rangle\big| \leq \frac{\kappa_{\mathcal{L}}}{2}\sum_\alpha \|\Delta_\alpha\|_2^2,$$

*when the number of samples scales as $n \geq c\left(\frac{\binom{|I|}{2}\Lambda^2|I|}{\kappa_{\mathcal{L}}}\right)^2\left(\max_\alpha \Psi_\alpha(\overline{\mathcal{M}}_\alpha)^2 + \max\{\log p, \log n\}\right)$, for some constant $c$ that depends only on the distribution of $X$.*

Condition (D3) is the usual restricted eigenvalue condition which has been analyzed previously in "clean-structured" model estimation, so that we can directly appeal to previous results [10, 12] to show that it holds with high probability when the observation matrix is drawn from the $\Sigma$-Gaussian ensemble.

We are now ready to derive the consequences of the deterministic bound in Theorem 1 for the case of the linear regression model above.

### 4.2 Linear Regression with Sparse and Group-sparse structures

We now consider the following superposition structure, comprised of both sparse and group-sparse structures. Suppose that a set of groups $\mathcal{G} = \{G_1, G_2, \ldots, G_q\}$ are disjoint subsets of the index-set $\{1, \ldots, p\}$, each of size at most $|G_i| \leq m$. Suppose that the linear regression parameter $\theta^*$ is a superposition of a group-sparse component $\theta_g^*$ with respect to this set of groups $\mathcal{G}$, as well as a sparse component $\theta_s^*$ with respect to the remaining indices $\{1, \ldots, p\}\backslash\cup_{i=1}^q G_i$, so that $\theta^* = \theta_g^* + \theta_s^*$. Then, we use the hybrid regularization function $\sum_{\alpha\in I}\lambda_\alpha \mathcal{R}_\alpha(\theta_\alpha) = \lambda_s\|\theta_s\|_1 + \lambda_g\|\theta_g\|_{1,a}$ where $\|\theta\|_{1,a} := \sum_{t=1}^q \|\theta_{G_t}\|_a$ for $a \geq 2$.

**Corollary 2.** *Consider the linear model (4) where $\theta^*$ is the sum of exact $s$-sparse $\theta_s^*$ and exact $s_g$ group-sparse $\theta_g^*$. Suppose that each row $X_i$ of the observation matrix $X$ is independently sampled from $N(0, \Sigma)$. Further, suppose that (6) holds and $w$ is sub-Gaussian with parameter $\sigma$. Then, if we solve (5) with*

$$\lambda_s = 8\sigma\sqrt{\frac{\log p}{n}} \quad and \quad \lambda_g = 8\sigma\left\{\frac{m^{1-1/a}}{\sqrt{n}} + \sqrt{\frac{\log q}{n}}\right\},$$

*then, with probability at least $1 - c_1\exp(-c_2 n\lambda_s^2) - c_3/q^2$, we have the error bound:*

$$\|\widehat{\theta} - \theta^*\|_2 \leq \frac{24\sigma}{\bar{\kappa}}\max\left\{\sqrt{\frac{s\log p}{n}}, \frac{\sqrt{s_g}m^{1-1/a}}{\sqrt{n}} + \sqrt{\frac{s_g\log q}{n}}\right\}.$$

Let us briefly compare the result from Corollary 2 with those from single-structured regularized estimators. Since the total sparsity of $\theta^*$ is bounded by $\|\theta\|_0 \leq ms_g + s$, "clean" $\ell_1$ regularized least squares, with high probability, gives the bound [11]: $\|\widehat{\theta}_{\ell_1} - \theta^*\|_2 = O\left(\sqrt{\frac{(ms_g+s)\log p}{n}}\right)$. On the other hand, the support of $\theta^*$ also can be interpreted as comprising $s_g + s$ disjoint groups in the worst case, so that "clean" $\ell_1/\ell_2$ group regularization entails, with high probability, the bound [11]: $\|\widehat{\theta}_{\ell_1/\ell_2} - \theta^*\|_2 = O\left(\sqrt{\frac{(s_g+s)m}{n}} + \sqrt{\frac{(s_g+s)\log q}{n}}\right)$. We can easily verify that Corollary 2 achieves better bounds, considering the fact $p \leq mq$.

## 5 Convergence Rates for Multiple Regression

In this section, we consider the multiple linear regression model, with $m$ linear regressions written jointly as

$$Y = X\Theta^* + W, \tag{7}$$

where $Y \in \mathbb{R}^{n \times m}$ is the observation matrix: with each column corresponding to a separate linear regression task, and $\Theta^* \in \mathbb{R}^{p \times m}$ is the collated set of parameters. $X \in \mathbb{R}^{n \times p}$ is the "observation" matrix; while $W \in \mathbb{R}^{n \times m}$ is collated set of observation noise vectors. For this class of statistical models, we will consider the instantiation of (1) with the loss function $\mathcal{L}$ consisting of the squared loss:

$$\min_{(\Theta_\alpha)_{\alpha \in I}} \left\{ \frac{1}{n} \|Y - X\big(\sum_{\alpha \in I} \Theta_\alpha\big)\|_F^2 + \sum_{\alpha \in I} \lambda_\alpha \mathcal{R}_\alpha(\Theta_\alpha) \right\}. \tag{8}$$

In contrast to the linear regression model in the previous section, the model (7) has a *matrix-structured* parameter; nonetheless conditions (C3-C4) in Theorem 1 reduce to the following conditions that are very similar to those in the previous section, with the Frobenius norm replacing the $\ell_2$ norm:

**(D3)** $\frac{1}{n}\|X\Delta_\alpha\|_F^2 \geq \kappa_{\mathcal{L}}\|\Delta_\alpha\|^2 - g_\alpha \mathcal{R}_\alpha^2(\Delta_\alpha)$ for all $\Delta_\alpha \in \Omega_\alpha$.

**(D4)** $\frac{2}{n}\left|\sum_{\alpha<\beta}\langle\!\langle X\Delta_\alpha, X\Delta_\beta\rangle\!\rangle\right| \leq \frac{\kappa_{\mathcal{L}}}{2}\sum_{\alpha \in I}\|\Delta_\alpha\|^2 + \sum_{\alpha \in I} h_\alpha \mathcal{R}_\alpha^2(\Delta_\alpha)$.

where the notation $\langle\!\langle A, B\rangle\!\rangle$ denotes the trace inner product, $\text{trace}(A^\top B) = \sum_i \sum_j A_{ij}B_{ij}$.

As in the previous linear regression example, we again impose the assumption (C-Linear) on the population covariance matrix of a $\Sigma$-Gaussian ensemble, but in this case with the notational change of $\mathcal{P}_{\bar{\mathcal{M}}_\alpha}$ denoting the matrix corresponding to projection operator onto the *row-spaces* of matrices in $\bar{\mathcal{M}}_\alpha$. Thus, with the low-rank matrix structure discussed in Section 2, we would have $\mathcal{P}_{\bar{\mathcal{M}}_\alpha} = UU^\top$. Under the (C-Linear) assumption, the following proposition then extends Proposition 2:

**Proposition 3.** *Consider the problem* (8) *with the matrix parameter* $\Theta$. *Under the same assumptions as in Proposition 2, we have with probability at least* $1 - \frac{4}{\max\{n,p\}}$,

$$\frac{2}{n}\left|\sum_{\alpha<\beta}\langle\!\langle X\Delta_\alpha, X\Delta_\beta\rangle\!\rangle\right| \leq \frac{\kappa_{\mathcal{L}}}{2}\sum_\alpha \|\Delta_\alpha\|_F^2.$$

Consider an instance of this multiple linear regression model with the superposition structure consisting of row-sparse, column-sparse and elementwise sparse matrices: $\Theta^* = \Theta_r^* + \Theta_c^* + \Theta_s^*$. In order to obtain estimators for this model, we use the hybrid regularization function $\sum_{\alpha \in I} \lambda_\alpha \mathcal{R}_\alpha(\theta_\alpha) = \lambda_r\|\Theta_r\|_{r,a} + \lambda_c\|\Theta_c\|_{c,a} + \lambda_s\|\Theta_s\|_1$ where $\|\cdot\|_{r,a}$ denotes the sum of $\ell_a$ norm of rows for $a \geq 2$, and similarly $\|\cdot\|_{c,a}$ is the sum of $\ell_a$ norm of columns, and $\|\cdot\|_1$ is entrywise $\ell_1$ norm for matrix.

**Corollary 3.** *Consider the multiple linear regression model* (7) *where* $\Theta^*$ *is the sum of* $\Theta_r^*$ *with* $s_r$ *nonzero rows,* $\Theta_c^*$ *with* $s_c$ *nonzero columns, and* $\Theta_s^*$ *with* $s$ *nonzero elements. Suppose that the design matrix* $X$ *is* $\Sigma$-*Gaussian ensemble with the properties of column normalization and* $\sigma_{\max}(X) \leq \sqrt{n}$. *Further, suppose that* (6) *holds and* $W$ *is elementwise sub-Gaussian with parameter* $\sigma$. *Then, if we solve* (8) *with*

$$\lambda_s = 8\sigma\sqrt{\frac{\log p + \log m}{n}}, \quad \lambda_r = 8\sigma\left\{\frac{m^{1-1/a}}{\sqrt{n}} + \sqrt{\frac{\log p}{n}}\right\}, \text{ and } \lambda_c = 8\sigma\left\{\frac{p^{1-1/a}}{\sqrt{n}} + \sqrt{\frac{\log m}{n}}\right\},$$

with probability at least $1 - c_1 \exp(-c_2 n \lambda_s^2) - \frac{c_3}{p^2} - \frac{c_3}{m^2}$, the error of the estimate $\widehat{\Theta}$ is bounded as:

$$\|\widehat{\Theta} - \Theta^*\|_2 \leq \frac{36\sigma}{\bar{\kappa}} \max\left\{ \sqrt{\frac{s(\log p + \log m)}{n}}, \ \frac{\sqrt{s_r} m^{1-1/a}}{\sqrt{n}} + \sqrt{\frac{s_r \log p}{n}}, \ \frac{\sqrt{s_c} p^{1-1/a}}{\sqrt{n}} + \sqrt{\frac{s_c \log m}{n}} \right\}.$$

# 6 Convergence Rates for Principal Component Analysis

In this section, we consider the robust/noisy principal component analysis problem, where we are given $n$ i.i.d. random vectors $Z_i \in \mathbb{R}^p$ where $Z_i = U_i + v_i$. $U_i \sim N(0, \Theta^*)$ is the "uncorrupted" set of observations, with a low-rank covariance matrix $\Theta^* = LL^T$, for some loading matrix $L \in \mathbb{R}^{p \times r}$. $v_i \in \mathbb{R}^p$ is a noise/error vector; in standard factor analysis, $v_i$ is a spherical Gaussian noise vector: $v_i \sim N(0, \sigma^2 I_{p \times p})$ (or $v_i = 0$); and the goal is to recover the loading matrix $L$ from samples.

In PCA with sparse noise, $v_i \sim N(0, \Gamma^*)$, where $\Gamma^*$ is elementwise sparse. In this case, the covariance matrix of $Z_i$ has the form $\Sigma = \Theta^* + \Gamma^*$, where $\Theta^*$ is low-rank, and $\Gamma^*$ is sparse. We can thus write the sample covariance model as: $Y := \frac{1}{n} \sum_{i=1}^{n} Z_i Z_i^T = \Theta^* + \Gamma^* + W$, where $W \in \mathbb{R}^{p \times p}$ is a Wishart distributed random matrix. For this class of statistical models, we will consider the following instantiation of (1):

$$\min_{(\Theta, \Gamma)} \left\{ \|Y - \Theta - \Gamma\|_F^2 + \lambda_\Theta \|\Theta\|_1 + \lambda_\Gamma \|\Gamma\|_1 \right\}. \tag{9}$$

where $\|\cdot\|_1$ denotes the nuclear norm while $\|\cdot\|_1$ does the element-wise $\ell_1$ norm (we will use $\|\cdot\|_2$ for the spectral norm.).

In contrast to the previous two examples, (9) includes a trivial design matrix, $X = I_{p \times p}$, which allows (D4) to hold under the simpler (C-linear) condition: where $\Lambda$ is $\max_{\gamma_1, \gamma_2} \left\{ 2 + \frac{3\lambda_{\gamma_1} \Psi_{\gamma_1}(\bar{\mathcal{M}}_{\gamma_1})}{\lambda_{\gamma_2} \Psi_{\gamma_2}(\bar{\mathcal{M}}_{\gamma_2})} \right\}$,

$$\max\left\{ \sigma_{\max}\left( \mathcal{P}_{\bar{\mathcal{M}}_\Theta} \mathcal{P}_{\bar{\mathcal{M}}_\Gamma} \right), \sigma_{\max}\left( \mathcal{P}_{\bar{\mathcal{M}}_\Theta} \mathcal{P}_{\bar{\mathcal{M}}_\Gamma^\perp} \right), \sigma_{\max}\left( \mathcal{P}_{\bar{\mathcal{M}}_\Theta^\perp} \mathcal{P}_{\bar{\mathcal{M}}_\Gamma} \right), \sigma_{\max}\left( \mathcal{P}_{\bar{\mathcal{M}}_\Theta^\perp} \mathcal{P}_{\bar{\mathcal{M}}_\Gamma^\perp} \right) \right\} \leq \frac{1}{16\Lambda^2}. \tag{10}$$

**Corollary 4.** *Consider the principal component analysis model where $\Theta^*$ has the rank $r$ at most and $\Gamma^*$ has $s$ nonzero entries. Suppose that* (10) *holds. Then, given the choice of*

$$\lambda_\Theta = 16\sqrt{\|\Sigma\|_2}\sqrt{\frac{p}{n}}, \ \lambda_\Gamma = 32\rho(\Sigma)\sqrt{\frac{\log p}{n}},$$

*where $\rho(\Sigma) = \max_j \Sigma_{jj}$, the optimal error of* (9) *is bounded by*

$$\|\widehat{\Theta} - \Theta^*\|_2 \leq \frac{48}{\bar{\kappa}} \max\left\{ \sqrt{\|\Sigma\|_2}\sqrt{\frac{rp}{n}}, \ 2\rho(\Sigma)\sqrt{\frac{s \log p}{n}} \right\},$$

*with probability at least $1 - c_1 \exp(-c_2 \log p)$.*

**Remarks.** Agarwal et al. [1] also analyze this model, and propose to use the $M$-estimator in (9), with the additional constraint of $\|\Theta\|_\infty \leq \frac{\alpha}{p}$. Under a stricter "global" RSC condition, they compute the error bound $\|\widehat{\Theta} - \Theta^*\|_2 \asymp \max\{\sqrt{\|\Sigma\|_2}\sqrt{\frac{rp}{n}}, \rho(\Sigma)\sqrt{\frac{s \log p}{n}} + \frac{\alpha}{p}\}$ where $\alpha$ is a parameter between $1$ and $p$. This bound is similar to that in Corollary 4, but with an additional term $\frac{\alpha}{p}$, so that it does not go to zero as a function of $n$. It also faces a trade-off: a smaller value of $\alpha$ to reduce error bound would make the assumption on the maximum element of $\Theta^*$ stronger as well. Our corollaries do not suffer these lacunae; see also our remarks in (R2) in Theorem 1. [14] extended the result of [1] to the special case where $\Theta^* = \Theta_r^* + \Theta_s^*$ using the notation of the previous section; the remarks above also apply here. Note that our work and [1] derive Frobenius error bounds under restricted strong convexity conditions; other recent works such as [7] also derive such Frobenius error bounds but under stronger conditions (see [1] for details).

### Acknowledgments

We acknowledge the support of ARO via W911NF-12-1-0390 and NSF via IIS-1149803, DMS-1264033.

# References

[1] A. Agarwal, S. Negahban, and M. J. Wainwright. Noisy matrix decomposition via convex relaxation: Optimal rates in high dimensions. *Annals of Statistics*, 40(2):1171–1197, 2012.

[2] E. J. Candès, J. K. Romberg, and T. Tao. Stable signal recovery from incomplete and inaccurate measurements. *Communications on Pure and Applied Mathematics*, 59(8):1207–1223, 2006.

[3] E. J. Candès, X. Li, Y. Ma, and J. Wright. Robust principal component analysis? *Journal of the ACM*, 58(3), May 2011.

[4] V. Chandrasekaran, B. Recht, P. A. Parrilo, and A. S. Willsky. The convex geometry of linear inverse problems. In *48th Annual Allerton Conference on Communication, Control and Computing*, 2010.

[5] V. Chandrasekaran, S. Sanghavi, P. A. Parrilo, and A. S. Willsky. Rank-sparsity incoherence for matrix decomposition. *SIAM Journal on Optimization*, 21(2), 2011.

[6] V. Chandrasekaran, P. A. Parrilo, and A. S. Willsky. Latent variable graphical model selection via convex optimization. *Annals of Statistics (with discussion)*, 40(4), 2012.

[7] D. Hsu, S. M. Kakade, and T. Zhang. Robust matrix decomposition with sparse corruptions. *IEEE Trans. Inform. Theory*, 57:7221–7234, 2011.

[8] A. Jalali, P. Ravikumar, S. Sanghavi, and C. Ruan. A dirty model for multi-task learning. In *Neur. Info. Proc. Sys. (NIPS)*, 23, 2010.

[9] M. McCoy and J. A. Tropp. Two proposals for robust pca using semidefinite programming. *Electron. J. Statist.*, 5:1123–1160, 2011.

[10] S. Negahban and M. J. Wainwright. Estimation of (near) low-rank matrices with noise and high-dimensional scaling. *Annals of Statistics*, 39(2):1069–1097, 2011.

[11] S. Negahban, P. Ravikumar, M. J. Wainwright, and B. Yu. A unified framework for high-dimensional analysis of M-estimators with decomposable regularizers. *Statistical Science*, 27 (4):538–557, 2012.

[12] G. Raskutti, M. J. Wainwright, and B. Yu. Restricted eigenvalue properties for correlated gaussian designs. *Journal of Machine Learning Research (JMLR)*, 99:2241–2259, 2010.

[13] R. Vershynin. Introduction to the non-asymptotic analysis of random matrices. In *Compressed Sensing: Theory and Applications*. Cambridge University Press, 2012.

[14] H. Xu and C. Leng. Robust multi-task regression with grossly corrupted observations. *Inter. Conf. on AI and Statistics (AISTATS)*, 2012.

[15] H. Xu, C. Caramanis, and S. Sanghavi. Robust pca via outlier pursuit. *IEEE Transactions on Information Theory*, 58(5):3047–3064, 2012.

